# On the Dirichlet Prior and Bayesian Regularization

**Harald Steck**
Artificial Intelligence Laboratory
Massachusetts Institute of Technology
Cambridge, MA 02139
*harald@ai.mit.edu*

**Tommi S. Jaakkola**
Artificial Intelligence Laboratory
Massachusetts Institute of Technology
Cambridge, MA 02139
*tommi@ai.mit.edu*

## Abstract

A common objective in learning a model from data is to recover its network *structure*, while the model parameters are of minor interest. For example, we may wish to recover regulatory networks from high-throughput data sources. In this paper we examine how Bayesian regularization using a product of independent Dirichlet priors over the model parameters affects the learned model *structure* in a domain with discrete variables. We show that a small scale parameter – often interpreted as "equivalent sample size" or "prior strength" – leads to a strong regularization of the model structure (sparse graph) given a sufficiently large data set. In particular, the *empty* graph is obtained in the limit of a vanishing scale parameter. This is diametrically opposite to what one may expect in this limit, namely the complete graph from an (unregularized) maximum likelihood estimate. Since the prior affects the parameters as expected, the scale parameter balances a *trade-off* between regularizing the parameters vs. the structure of the model. We demonstrate the benefits of optimizing this trade-off in the sense of predictive accuracy.

## 1 Introduction

Regularization is essential when learning from finite data sets. In the Bayesian approach, regularization is achieved by specifying a prior distribution over the parameters and subsequently *averaging* over the posterior distribution. This regularization provides not only smoother estimates of the parameters compared to maximum likelihood but also guides the selection of model structures.

It was pointed out in [6] that a very *large* scale parameter of the Dirichlet prior can degrade predictive accuracy due to severe regularization of the *parameter* estimates. We complement this discussion here and show that a very *small* scale parameter can lead to poor over-regularized *structures* when a product of (conjugate) *Dirichlet priors* is used over multinomial conditional distributions (Section 3). Section 4 demonstrates the effect of the scale parameter and how it can be calibrated. We focus on the class of Bayesian network models throughout this paper.

## 2 Regularization of Parameters

We briefly review Bayesian regularization of *parameters*. We follow the assumptions outlined in [6]: multinomial sample, complete data, parameter modularity, parameter independence, and Dirichlet prior. Note that the Dirichlet prior over the parameters is often used for two reasons: (1) the conjugate prior permits analytical calculations, and (2) the Dirichlet prior is intimately tied to the desirable likelihood-equivalence property of network structures [6]. The Dirichlet prior over the parameters $\theta_{\cdot|\pi_i}$ is given by

$$p(\theta_{X_i|\pi_i}) = \frac{\Gamma(\sum_{x_i} \alpha_{x_i,\pi_i})}{\prod_{x_i} \Gamma(\alpha_{x_i,\pi_i})} \prod_{x_i} \theta_{x_i|\pi_i}^{\alpha_{x_i,\pi_i}-1}, \tag{1}$$

where $\theta_{x_i|\pi_i}$ pertains to variable $X_i$ in state $x_i$ given that its parents $\Pi_i$ are in joint state $\pi_i$. The number of variables in the domain is denoted by $n$, and $i = 1, ..., n$. The normalization terms in Eq. 1 involve the Gamma function $\Gamma(\cdot)$. There are a number of approaches to specifying the positive hyper-parameters $\alpha_{x_i,\pi_i}$ of the Dirichlet prior [2, 1, 6]; we adopt the common choice,

$$\alpha_{x_i,\pi_i} = \alpha \cdot p(x_i, \pi_i), \tag{2}$$

where $p$ is a (marginal) prior distribution over the (joint) states, as this assignment ensures likelihood equivalence of the network structures [6]. Due to lack of prior knowledge, $p$ is often chosen to be uniform, $p(x_i, \pi_i) = 1/(|X_i| \cdot |\Pi_i|)$, where $|X_i|$, $|\Pi_i|$ denote the number of (joint) states [1]. The *scale parameter* $\alpha$ of the Dirichlet prior is positive and independent of $i$, i.e., $\alpha = \sum_{x_i,\pi_i} \alpha_{x_i,\pi_i}$.

The *average* parameter value $\bar{\theta}_{x_i|\pi_i}$, which typically serves as the regularized parameter estimate given a network structure $m$, is given by

$$\bar{\theta}_{x_i|\pi_i} \equiv E_{p(\theta_{X_i|\pi_i}|D,m)}[\theta_{x_i|\pi_i}] = \frac{N_{x_i,\pi_i} + \alpha_{x_i,\pi_i}}{N_{\pi_i} + \alpha_{\pi_i}}, \tag{3}$$

where $N_{x_i,\pi_i}$ are the cell-counts from data $D$; $E[\cdot]$ is the expectation. Positive hyper-parameters $\alpha_{x_i,\pi_i}$ lead to regularized parameter estimates, i.e., the estimated parameters become "smoother" or "less extreme" when the prior distribution $p$ is close to uniform. An increasing scale parameter $\alpha$ leads to a stronger regularization, while in the limit $\alpha \to 0$, the (unregularized) maximum likelihood estimate is obtained, as expected.

## 3 Regularization of Structure

In the remainder of this paper, we outline effects due to Bayesian regularization of the Bayesian network structure when using a product of Dirichlet priors. Let us briefly introduce relevant notation.

In the Bayesian approach to structure learning, the posterior probability of the network structure $m$ is given by $p(m|D) = p(D|m)p(m)/p(D)$, where $p(D)$ is the (unknown) probability of given data $D$, and $p(m)$ denotes the prior distribution over the network structures; we assume $p(m) > 0$ for all $m$. Following the assumptions outlined in [6], including the Dirichlet prior over the parameters $\theta$, the marginal likelihood $p(D|m) = E_{p(\theta|m)}[p(D|m,\theta)]$ can be calculated analytically. Pretending that the (i.i.d.) data arrived in a sequential manner, it can be written as

$$p(D|m) = \prod_{k=1}^{N} \prod_{i=1}^{n} \frac{N_{x_i^k,\pi_i^k}^{(k-1)} + \alpha_{x_i^k,\pi_i^k}}{N_{\pi_i^k}^{(k-1)} + \alpha_{\pi_i^k}}, \tag{4}$$

where $N^{(k-1)}$ denotes the counts implied by data $D^{(k-1)}$ seen *before* step $k$ along the sequence ($k = 1, ..., N$). The (joint) state of variable $X_i$ and its parents $\Pi_i$ occurring in the $k^{\text{th}}$ data point is denoted by $x_i^k, \pi_i^k$. In Eq. 4, we also decomposed the joint probability into a product of conditional probabilities according to the Bayesian network structure $m$. Eq. 4 is independent of the sequential ordering of the data points, and the ratio in Eq. 3 corresponds to the one in Eq. 4 when based on data $D^{(k-1)}$ at each step $k$ along the sequence.

## 3.1 Limit of Vanishing Scale-Parameter

This section is concerned with the limit of a vanishing scale parameter of the Dirichlet prior, $\alpha \to 0$. In this limit Bayesian regularization depends crucially on the number of zero-cell-counts in the contingency table implied by the data, or in other words, on the number of *different* configurations (data points) contained in the data. Let the *Effective Number of Parameters* (EP) be defined as

$$d_{\text{EP}}^{(m)} = \sum_{i=1}^{n} [\sum_{x_i, \pi_i} I(N_{x_i, \pi_i}) - \sum_{\pi_i} I(N_{\pi_i})], \qquad (5)$$

where $N_{x_i, \pi_i}$, $N_{\pi_i}$ are the (marginal) cell counts in the contingency table implied by data $D$; $m$ refers to the Bayesian network structure, and $I(\cdot)$ is an indicator function such that $I(z) = 0$ if $z = 0$ and $I(z) = 1$ otherwise. When all cell counts are positive, EP is identical to the well-known *number of parameters* (P), $d_{\text{EP}}^{(m)} = d_{\text{P}}^{(m)} = \sum_i (|X_i| - 1)|\Pi_i|$, which play an important role in regularizing the learned network structure. The key difference is that EP accounts for zero-cell-counts implied by the data.

Let us now consider the behavior of the marginal likelihood (cf. Eq. 4) in the limit of a small scale parameter $\alpha$. We find

**Proposition 1:** *Under the assumptions concerning the prior distribution outlined in Section 2, the marginal likelihood of a Bayesian network structure m vanishes in the limit $\alpha \to 0$ if the data D contain two or more different configurations. This property is independent of the network structure. The leading polynomial order is given by*

$$p(D|m) \sim \alpha^{d_{\text{EP}}^{(m)}} \qquad \text{as } \alpha \to 0, \qquad (6)$$

*which depends both on the network structure and the data. However, the dependence on the data is through the number of* different *data points only. This holds independently of a particular choice of strictly positive prior distributions $p(X_i, \Pi_i)$. If the prior over the network structures is strictly positive, this limiting behavior also holds for the posterior probability $p(m|D)$.*

In the following we give a derivation of Proposition 1 that also facilitates the intuitive understanding of the result. First, let us consider the behavior of the *Dirichlet* distribution in the limit $\alpha \to 0$. The hyper-parameters $\alpha_{x_i, \pi_i}$ vanish when $\alpha \to 0$, and thus the Dirichlet prior converges to a discrete distribution over the parameter simplex in the sense that the probability mass concentrates at a particular, randomly chosen corner of the simplex containing $\theta_{\cdot|\pi_i}$ (cf. [9]). Since the randomly chosen points (for different $\pi_i, i$) do not change when sampling (several) data points from the distribution implied by the model, it follows immediately that the marginal likelihood of any network structure vanishes whenever there are two or more different configurations contained in the data.

This well-known fact also shows that the limit $\alpha \to 0$ actually corresponds to a very *strong* prior belief [9, 12]. This is in contrast to many traditional interpretations where the limit $\alpha \to 0$ is considered as "no prior information", often motivated by Eq. 3. As pointed out in [9, 12], the interpretation of the scale parameter $\alpha$

as "equivalent sample size" or as the "strength" of prior belief may be misleading, particularly in the case where $\alpha_{x_i,\pi_i} < 1$ for some configurations $x_i, \pi_i$. A review of different notions of "noninformative" priors (including their limitations) can be found in [7]. Note that the noninformative prior in the sense of entropy is achieved by setting $\alpha_{x_i,\pi_i} = 1$ for each $x_i, \pi_i$ and for all $i = 1,...,n$. This is the assignment originally proposed in [2]; however, this assignment generally is inconsistent with Eq. 2, and hence with likelihood equivalence [6].

In order to explain the behavior of the marginal likelihood in leading order of the scale parameter $\alpha$, the properties of the Dirichlet distribution are not sufficient by themselves. Additionally, it is essential that the probability distribution described by a Bayesian network decomposes into a *product* of *conditional* probabilities, and that there is a Dirichlet prior pertaining to each variable for *each* parent *configuration*. All these Dirichlet priors are independent of each other under the standard assumption of parameter independence. Obviously, the ratio (for given $k$ and $i$) in Eq. 4 can only vanish in the limit $\alpha \to 0$ if $N_{x_i^k,\pi_i^k}^{(k-1)} = 0$ while $N_{\pi_i^k}^{(k-1)} > 0$; in other words, the parent-configuration $\pi_i^k$ must already have occurred previously along the sequence ($\pi_i^k$ is "old"), while the child-state $x_i^k$ occurs simultaneously with this parent-state for the *first* time ($x_i^k, \pi_i^k$ is "new"). In this case, the leading polynomial order of the ratio (for given $k$ and $i$) is linear in $\alpha$, assuming $p(X_i, \Pi_i) > 0$; otherwise the ratio (for given $k$ and $i$) converges to a finite positive value in the limit $\alpha \to 0$. Consequently, the dependence of the marginal likelihood in leading polynomial order on $\alpha$ is completely determined by the number of *different* configurations in the data. It follows immediately that the leading polynomial order in $\alpha$ is given by EP (cf. Eq. 5). This is because the first term counts the number of all the different joint configurations of $X_i, \Pi_i$ in the data, while the second term ensures that EP counts only those configurations where $(x_i^k, \pi_i^k)$ is "new" while $\pi_i^k$ is "old".

Note that the behavior of the marginal likelihood in Proposition 1 is not entirely determined by the network structure in the limit $\alpha \to 0$, as it still depends on the data. This is illustrated in the following example. First, let us consider two binary variables, $X_0$ and $X_1$, and the data $D$ containing only two data points, say $(0,0)$ and $(1,1)$. Given data $D$, three Dirichlet priors are relevant regarding graph $m_1$, $X_0 \to X_1$, but only two Dirichlet priors pertain to the empty graph, $m_0$. The resulting additional "flexibility" due to an increased *number* of priors favours more complex models: $p(D|m_1) \sim \alpha$, while $p(D|m_0) \sim \alpha^2$. Second, let us now assume that all possible configurations occur in data $D$. Then we still have $p(D|m_0) \sim \alpha^2$ for the empty graph. Concerning graph $m_1$, however, the marginal likelihood now also involves the *vanishing* terms due to the two priors pertaining to $\theta_{X_1|X_0=0}$ and $\theta_{X_1|X_0=1}$, and it hence becomes $p(D|m_1) \sim \alpha^3$.

This dependence on the data can be formalized as follows. Let us compare the marginal likelihoods of two graphs, say $m^+$ and $m^-$. In particular, let us consider two graphs that are identical except for a single edge, say $A \leftarrow B$ between the variables $A$ and $B$. Let the edge be present in graph $m^+$ and absent in $m^-$. The fact that the marginal likelihood decomposes into terms pertaining to each of the variables (cf. Eq. 4) entails that all the terms regarding the remaining variables cancel out in the Bayes factor $p(D|m^+)/p(D|m^-)$, which is the standard *relative* Bayesian score. With the definition of the *Effective Degrees of Freedom* (EDF)[1]

$$d_{\mathrm{EDF}} = d_{\mathrm{EP}}^{(m^+)} - d_{\mathrm{EP}}^{(m^-)}, \tag{7}$$

we immediately obtain from Proposition 1 that $p(D|m^+)/p(D|m^-) \sim \alpha^{d_{\mathrm{EDF}}}$ in the

limit $\alpha \to 0$, and hence

**Proposition 2:** *Let $m^+$ and $m^-$ be the two network structures as defined above. Let the prior belief be given according to Eq. 2. Then in the limit $\alpha \to 0$:*

$$\log \frac{p(D|m^+)}{p(D|m^-)} \to \begin{cases} -\infty & \text{if } d_{\text{EDF}} > 0, \\ +\infty & \text{if } d_{\text{EDF}} < 0. \end{cases} \tag{8}$$

*The result holds independently of a particular choice of strictly positive prior distributions $p(X_i, \Pi_i)$. If the prior over the network structures is strictly positive, this limiting behavior also holds for the posterior ratio.*

A positive value of the log Bayes factor indicates that the presence of the edge $A \leftarrow B$ is favored, given the parents $\Pi_A$; conversely, a negative relative score suggests that the absence of this edge is preferred. The divergence of this relative Bayesian score implies that there exists a (small) positive threshold value $\alpha_O > 0$ such that, for any $\alpha < \alpha_O$, the same graph(s) are favored as in the limit.

Since Proposition 2 applies to every edge in the network, it follows immediately that the *empty (complete)* graph is assigned the highest relative Bayesian score when EDF are positive (negative). Regularization of network structure in the case of positive EDF is therefore extreme, permitting only the empty graph. This is precisely the opposite of what one may have expected in this limit, namely the complete graph corresponding to the *unregularized* maximum likelihood estimate (MLE). In contrast, when EDF are negative, the complete graph is favored. This agrees with MLE.

Roughly speaking, positive (negative) EDF correspond to large (small) data sets. It is thus surprising that a small data set, where one might expect an increased restriction on model complexity, actually gives rise to the complete graph, while a large data set yields the – most regularized – empty graph in the limit $\alpha \to 0$. Moreover, it is conceivable that a "medium" sized data set may give rise to *both* positive and negative EDF. This is because the *marginal* contingency tables implied by the data with respect to a sparse (dense) graph may contain a small (large) number of zero-cell-counts. The relative Bayesian score can hence become rather unstable in this case, as completely different graph structures are optimal in the limit $\alpha \to 0$, namely graphs where each variable has either the maximal number of parents or none.

Note that there are two reasons for the hyper-parameters $\alpha_{x_i, \pi_i}$ to take on small values (cf. Eq. 2): (1) a small equivalent sample size $\alpha$, or (2) a large number of joint states, i.e. $|X_i| \cdot |\Pi_i| \gg \alpha$, due to a large number of parents (with a large number of states). Thus, these hyper-parameters can also vanish in the limit of a large number of configurations $(x, \pi)$ even though the scale parameter $\alpha$ remains fixed. This is precisely the limit defining Dirichlet processes [4], which, analogously, produce discrete samples. With a finite data set and a large number of joint configurations, only the typical limit in Proposition 2 is possible. This follows from the fact that a large number of zero-cell-counts forces EDF to be negative. The surprising behavior implied by Proposition 2 therefore does not carry over to Dirichlet processes. As found in [8], however, the use of a product of Dirichlet process priors in nonparametric inference can also lead to surprising effects.

When $d_{\text{EDF}} = 0$, it is indeed true that the value of the log Bayes factor can converge to any (possibly finite) value as $\alpha \to 0$. Its value is determined by the priors $p(X_i, \Pi_i)$, as well as by the counts implied by the data. The value of the Bayes factor can be therefore easily set by adjusting the prior weights $p(x_i, \pi_i)$.

## 3.2 Large Scale-Parameter

In the other limiting case, where $\alpha \to \infty$, the Bayes factor approaches a finite value, which in general depends on the given data and on the prior distributions $p(X_i, \Pi_i)$.

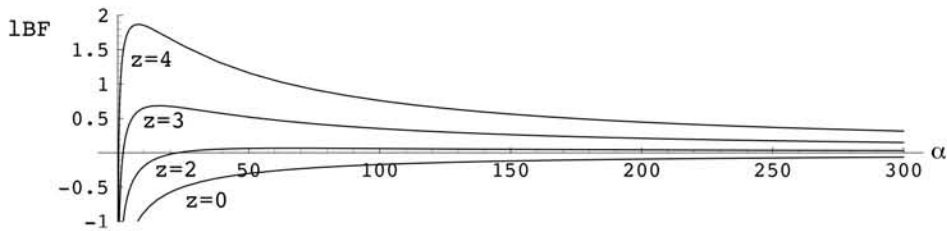

Figure 1: The log Bayes factor (lBF) is depicted as a function of the scale parameter $\alpha$. It is assumed that the two variables $A$ and $B$ are binary and have no parents; and that the "data" imply the contingency table: $N_{A=0,B=0} = N_{A=1,B=1} = 10 + z$ and $N_{A=1,B=0} = N_{A=0,B=1} = 10 - z$, where $z$ is a free parameter determining the statistical dependence between $A$ and $B$. The prior $p(X_i, \Pi_i)$ was chosen to be uniform.

This can be seen easily by applying the Stirling approximation in the limit $\alpha \to \infty$ after rewriting Eq. 4 in terms of Gamma functions (cf. also [2, 6]). When the popular choice of a uniform prior $p(X_i, \Pi_i)$ is used [1], then

$$\log \frac{p(D|m^+)}{p(D|m^-)} \to 0 \quad \text{as} \quad \alpha \to \infty, \tag{9}$$

which is independent of the data. Hence, neither the presence nor the absence of the edge between $A$ and $B$ is favored in this limit. Given a uniform prior over the network structures, $p(m) =$const, the posterior distribution $p(m|D)$ over the graphs thus becomes increasingly spread out as $\alpha$ grows, permitting more variable network structures.

The behavior of the Bayes factor between the two limits $\alpha \to 0$ and $\alpha \to \infty$ is exemplified for positive EDF in Figure 1: there are two qualitatively different behaviors, depending on the degree of statistical dependence between $A$ and $B$. A sufficiently weak dependence results in a monotonically increasing Bayes factor which favors the absence of the edge $A \leftarrow B$ at any finite value of $\alpha$. In contrast, given a sufficiently strong dependence between $A$ and $B$, the log Bayes factor takes on positive values for all (finite) $\alpha$ exceeding a certain value $\alpha_+$ of the scale parameter. Moreover, $\alpha_+$ grows as the statistical dependence between $A$ and $B$ diminishes. *Consequently, given a domain with a range of degrees of statistical dependences, the number of edges in the learned graph* increases *monotonically with growing scale parameter $\alpha$ when each variable has at most one parent (i.e., in the class of trees or forests).* This is because increasingly weaker statistical dependencies between variables are recovered as $\alpha$ grows; the restriction to forests excludes possible "interactions" among (several) parents of a variable. As suggested by our experiments, this increase in the number of edges can also be expected to hold for general Bayesian network structures (although not necessarily in a monotonic way).

This reveals that regularization of network structure tends to *diminish* with a growing scale parameter. Note that this is in the *opposite* direction to the regularization of parameters (cf. Section 2). *Hence, the scale parameter $\alpha$ of the Dirichlet prior determines the* trade-off *between regularizing the parameters vs. the structure of the Bayesian network model.*

If a uniform prior over the network structures is chosen, $p(m) =$const, the above discussion also holds for the posterior ratio (instead of the Bayes factor). The behavior is more complicated, however, when a non-uniform prior is assumed. For instance, when a prior is chosen that penalizes the presence of edges, the posterior

favours the absence of an edge not only when the scale parameter is sufficiently small, but also when it is sufficiently large. This is apparent from Fig. 1, when the log Bayes factor is compared to a *positive* threshold value (instead of zero).

## 4  Example

This section exemplifies that the *entire* model (parameters *and* structure) has to be considered when learning from data. This is because regularization of model structure diminishes, while regularization of parameters increases with a growing scale parameter $\alpha$ of the Dirichlet prior, as discussed in the previous sections.

When the entire model is taken into account, one can use a sensitivity analysis in order to determine the dependence of the learned model on the scale parameter $\alpha$, given the prior $p(X_i, \Pi_i)$ (cf. Eq. 2). The influence of the scale parameter $\alpha$ on *predictive* accuracy of the model can be assessed by cross-validation or, in a Bayesian approach, prequential validation [11, 3]. Another possibility is to treat the scale parameter $\alpha$ as an additional parameter of the model to be learned from data. Hence, prior belief regarding the parameters $\theta$ can then enter only through the (normalized) distributions $p(X_i, \Pi_i)$. However, note that this is sufficient to determine the (*average*) *prior* parameter estimate $\bar{\theta}$ (cf. Eq. 3), i.e., when $N = 0$. Assuming an (improper) uniform prior distribution over $\alpha$, its posterior distribution is $p(\alpha|D) \propto p(D|\alpha)$, given data $D$. Then $\alpha_D = \text{argmax}_\alpha p(D|\alpha)$, where $p(D|\alpha) = \sum_m p(D|\alpha, m) \, p(m)^2$ can be calculated exactly if the summation is feasible (like in the example below). Alternatively, assuming that the posterior over $\alpha$ is strongly peaked, the likelihood may also be approximated by summing over the $k$ most likely graphs $m$ only ($k = 1$ in the most extreme case; empirical Bayes). Subsequently, model structure $m$ and parameters $\bar{\theta}$ can be learned with respect to the Bayesian score employing $\alpha_D$.

In the following, the effect of various values assigned to the scale parameter $\alpha$ is exemplified concerning the data set gathered from Wisconsin high-school students by Sewell and Shah [10]. This domain comprises 5 discrete variables, each with 2 or 4 states; the sample size is 10,318. In this small domain, *exhaustive* search in the space of Bayesian network structures is feasible (29, 281 graphs). Both the prior distributions $p(m)$ for all $m$ and $p(X_i, \Pi_i)$ are chosen to be uniform. Figure 2 shows that the number of edges in the graph with the highest posterior probability grows with an increasing value of the scale parameter, as expected (cf. Section 3). In addition, cross-validation indicates best predictive accuracy of the learned model at $\alpha \approx 100, ..., 300$, while the likelihood $p(D|\alpha)$ takes on its maximum at $\alpha_D \approx 69$. Both approaches agree on the same network structure, which is depicted in Fig. 3. This graph can easily be interpreted in a causal manner, as outlined in [5].[3] We note that this graph was also obtained in [5] due, however, to additional constraints concerning network structure, as a rather small prior strength of $\alpha = 5$ was used. For comparison, Fig. 3 also shows the highest-scoring unconstraint graph due to $\alpha = 5$, which does not permit a causal interpretation, cf. also [5]. This illustrates that the "right" choice of the scale parameter $\alpha$ of the Dirichlet prior, when accounting for both model structure and parameters, can have a crucial impact on the learned network *structure* and the resulting insight in the ("true") dependencies among the variables in the domain.

| $\alpha$ | a. | XV 5 | $\frac{p(D\|\alpha)}{p(D\|\alpha_D)}$ |
|---|---|---|---|
| 5 | 6 | 0.045 | $10^{-10}$ |
| 50 | 7 | 0.044 | 0.13 |
| 100 | 7 | 0.040 | 0.05 |
| 200 | 7 | 0.040 | $10^{-14}$ |
| 300 | 7 | 0.040 | $10^{-30}$ |
| 500 | 7 | 0.042 | $10^{-65}$ |
| 1,000 | 8 | 0.047 | $10^{-151}$ |

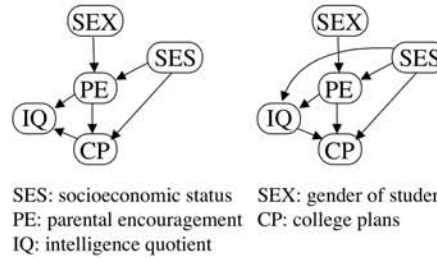

SES: socioeconomic status    SEX: gender of student
PE: parental encouragement   CP: college plans
IQ: intelligence quotient

Figure 2: As a function of $\alpha$: number of arcs (a.) in the highest-scoring graph; average KL divergence in 5-fold cross-validation (XV 5), std= 0.006; likelihood of $\alpha$ when treated as an additional model parameter ($\alpha_D = 69$).

Figure 3: Highest-scoring (unconstraint) graphs when $\alpha = 5$ (left), and when $\alpha = 46, ..., 522$ (right). Note that the latter graph can also be obtained at $\alpha = 5$ when additional constraints are imposed on the structure, cf. [5].

## Acknowledgments

We would like to thank Chen-Hsiang Yeang and the anonymous referees for valuable comments. Harald Steck acknowledges support from the German Research Foundation (DFG) under grant STE 1045/1-1. Tommi Jaakkola acknowledges support from Nippon Telegraph and Telephone Corporation, NSF ITR grant IIS-0085836, and from the Sloan Foundation in the form of the Sloan Research Fellowship.

## Footnotes

[1]Note that EDF are not necessarily non-negative.

[2]We assume that $m$ and $\alpha$ are independent a priori, $p(m|\alpha) = p(m)$.

[3]Since we did not impose any constraints on the network structure, unlike to [5], Markov-equivalence leaves the orientation of the edge between the variables IQ and CP unspecified.

## References

[1] W. Buntine. Theory refinement on Bayesian networks. *Conference on Uncertainty in Artificial Intelligence*, pages 52–60. Morgan Kaufmann, 1991.

[2] G. Cooper and E. Herskovits. A Bayesian method for the induction of probabilistic networks from data. *Machine Learning*, 9:309–47, 1992.

[3] A. P. Dawid. Statistical theory. The prequential approach. *Journal of the Royal Statistical Society, Series A*, 147:277–305, 1984.

[4] T. S. Ferguson. A Bayesian analysis of some nonparametric problems. *Annals of Statistics*, 1:209–30, 1973.

[5] D. Heckerman. A tutorial on learning with Bayesian networks. In M. I. Jordan (Ed.), *Learning in Graphical Models*, pages 301–54. Kluwer, 1996.

[6] D. Heckerman, D. Geiger, and D. M. Chickering. Learning Bayesian networks: the combination of knowledge and statistical data. *Machine Learning*, 20:197–243, 1995.

[7] R. E. Kass and L. Wasserman. Formal rules for selecting prior distributions: a review and annotated bibliography. Technical Report 583, CMU, 1993.

[8] S. Petrone and A. E. Raftery. A note on the Dirichlet process prior in Bayesian nonparametric inference with partial exchangeability. Technical Report 297, University of Washington, Seattle, 1995.

[9] J. Sethuraman and R. C. Tiwari. Convergence of Dirichlet measures and the interpretation of their parameter. In S. S. Gupta and J. O. Berger (Eds.), *Statistical Decision Theory and Related Topics III*, pages 305–15, 1982.

[10] W. Sewell and V. Shah. Social class, parental encouragement, and educational aspirations. *American Journal of Sociology*, 73:559–72, 1968.

[11] M. Stone. Cross-validatory choice and assessment of statistical predictions. *Journal of the Royal Statistical Society, Series B*, 36:111–47, 1974.

[12] S. G. Walker and B. K. Mallick. A note on the scale parameter of the Dirichlet process. *The Canadian Journal of Statistics*, 25:473–9, 1997.
